# Functional form of motion priors in human motion perception

**Hongjing Lu** [1,2]
hongjing@ucla.edu

**Tungyou Lin** [3]
tungyoul@math.ucla.edu

**Alan L. F. Lee** [1]
alanlee@ucla.edu

**Luminita Vese** [3]
lvese@math.ucla.edu

**Alan Yuille** [1,2,4]
yuille@stat.ucla.edu
Department of Psychology[1], Statistics[2], Mathematics[3] and Computer Science[4], UCLA

## Abstract

It has been speculated that the human motion system combines noisy measurements with prior expectations in an optimal, or rational, manner. The basic goal of our work is to discover experimentally which prior distribution is used. More specifically, we seek to infer the functional form of the motion prior from the performance of human subjects on motion estimation tasks. We restricted ourselves to priors which combine three terms for motion slowness, first-order smoothness, and second-order smoothness. We focused on two functional forms for prior distributions: L2-norm and L1-norm regularization corresponding to the Gaussian and Laplace distributions respectively. In our first experimental session we estimate the weights of the three terms for each functional form to maximize the fit to human performance. We then measured human performance for motion tasks and found that we obtained better fit for the L1-norm (Laplace) than for the L2-norm (Gaussian). We note that the L1-norm is also a better fit to the statistics of motion in natural environments. In addition, we found large weights for the second-order smoothness term, indicating the importance of high-order smoothness compared to slowness and lower-order smoothness. To validate our results further, we used the best fit models using the L1-norm to predict human performance in a second session with different experimental setups. Our results showed excellent agreement between human performance and model prediction – ranging from 3% to 8% for five human subjects over ten experimental conditions – and give further support that the human visual system uses an L1-norm (Laplace) prior.

## 1 Introduction

Imagine that you are traveling in a moving car and observe a walker through a fence full of punch holes. Your visual system can readily perceive the walking person against the apparently moving background using only the motion signals visible through these holes. But this task is far from trivial due to the inherent local ambiguity of motion stimuli, often referred to as the *aperture problem*. More precisely, if you view a line segment through an aperture then you can easily estimate the motion component normal to the line but it is impossible to estimate the tangential component. So there are an infinite number of possible interpretations of the local motion signal.

One way to overcome this local ambiguity is to integrate local motion measurements across space to infer the "true" motion field. Physiological studies have shown that direction-selective neurons

in primary visual cortex perform local measurements of motion. Then the visual system integrates these local motion measurements to form global motion perception [4, 5]. Psychophysicists have identified a variety of phenomena, such as motion capture and motion cooperativity, which appear to be consequences of motion spatial integration [1, 2, 3]. From the computational perspective, a number of Bayesian models have been proposed to explain these effects by hypothesizing prior assumptions about the motion fields that occur in natural environments. In particular, it has been shown that a prior which is biased to slow-and-smooth motion can account for a range of experimental results [6, 7, 8, 9, 10].

But although evidence from physiology and psychophysics supports the existence of an integration stage, it remains unclear exactly what motion priors are used to resolve the measurement ambiguities. In the walking example described above (see figure 1), the visual system needs to integrate the local measurements in the two regions within the red boxes in order to perceive a coherently moving background. This integration must be performed over large distances, because the regions are widely separated, but this integration cannot be extended to include the walker region highlighted in the blue box, because this would interfere with accurate estimation of the walker's movements. Hence the motion priors used by the human visual system must have a functional form which enables flexible and robust integration.

We aim to determine the functional form of the motion priors which underly human perception, and to validate how well these priors can influence human perception in various motion tasks. Our approach is to combine parametric modeling of the motion priors with psychophysical experiments to estimate the model parameters that provide the best fit to human performance across a range of stimulus conditions. To provide further validation, we then use the estimated model to predict human performance in several different experimental setups. In this paper, we first introduce the two functional forms which we consider and review related literature in Section 2. Then in Section 3 we present our computational theory and implementation details. In Section 4 we test the theory by comparing its predictions with human performance in a range of psychophysical experiments.

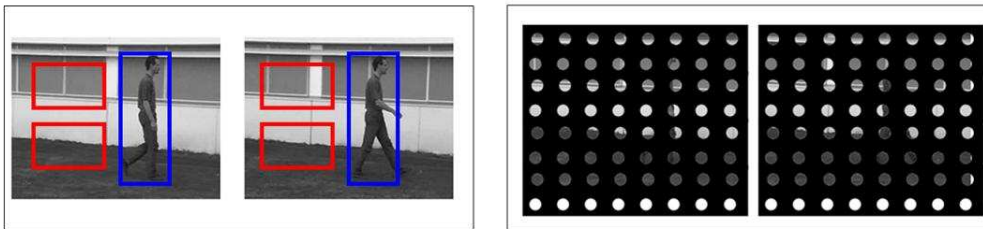

Figure 1: Observing a walker with a moving camera. Left panel, two example frames. The visual system needs to integrate motion measurements from the two regions in the red boxes in order to perceive the motion of the background. But this integration should not be extended to the walker region highlighted in the blue box. Right panel, the integration task is made harder by observing the scene through a set of punch holes. The experimental stimuli in our psychophysical experiments are designed to mimic these observation conditions.

## 2   Functional form of motion priors

Many models have proposed that the human visual system uses prior knowledge of probable motions, but the functional form for this prior remains unclear. For example, several well-established computational models employ Gaussian priors to encode the bias towards slow and spatially smooth motion fields. But the choice of Gaussian distributions has largely been based on computational convenience [6, 8], because they enable us to derive analytic solutions.

However, some evidence suggests that different distribution forms may be used by the human visual system. Researchers have used motion sequences in real scenes to measure the spatial and temporal statistics of motion fields [11, 12]. These natural statistics show that the magnitude of the motion (speed) falls off in a manner similar to a Laplacian distribution ( L1-norm regularization), which has heavier tails than Gaussian distributions (see the left plot in figure 2). These heavy tails indicates that while slow motions are very common, fast motions are still occur fairly frequently in natural

environments. A similar distribution pattern was also found for spatial derivatives of the motion flow, showing that non-smooth motion fields can also happen in natural environments. This statistical finding is not surprising since motion discontinuities can arise in the natural environment due to the relative motion of objects, foreground/background segmentation, and occlusion.

Stocker and Simoncelli [10] conducted a pioneering study to infer the functional form of the slowness motion prior. More specifically, they used human subject responses in a speed discrimination task to infer the shape of the slowness prior distribution. Their inferred slowness prior showed significantly heavier tails than a Gaussian distribution. They showed that a motion model using this inferred prior provided an adequate fit to human data for a wide range of stimuli.

Finally, the robustness of the L1-norm has also been demonstrated in many statistical applications (e.g., regression and feature selection). In the simplest case of linear regression, suppose we want to find the intercept with the constraint of zero slope. The regression with L1-norm regularization estimates the intercept based on the sample median, whereas the L2-norm regression estimates the intercept based on the sample mean. A single outlier has very little effect on the median but can alter the mean significantly. Accordingly, the L1-norm regularization is less sensitive to outliers than is the L2-norm. We illustrate this for motion estimation by the example in the right panel of figure 2. If there is a motion boundary in the true motion field, then a model using L2-norm regularization (Gaussian priors) tends to impose strong smoothing over the two distinct motion fields which blurs the motion across discontinuity. But the model with an L1-norm (Laplace prior) preserves the motion discontinuity and gives smooth motion flow on both sides of it.

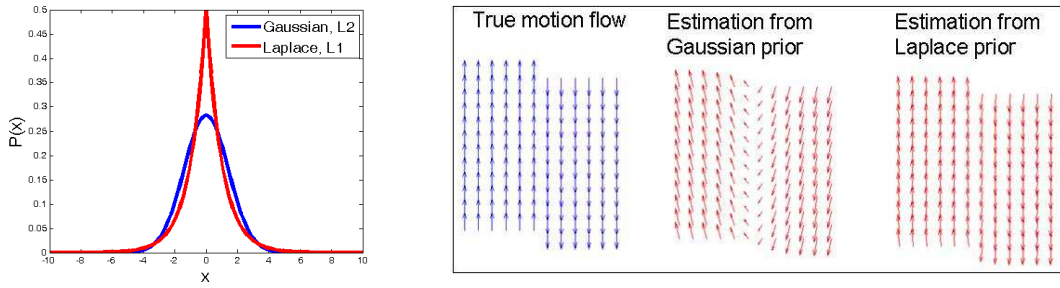

Figure 2: Left plot, the Gaussian distribution (L2-norm regularization) and the Laplace distribution (L1-norm regularization). Right plot, an illustration of over-smoothing caused by using Gaussian priors.

## 3  Mathematical Model

The input data is specified by local motion measurements $\vec{r}_q$, of form $\vec{u}_q = (u_{1q}, u_{2q})$, at a discrete set of positions $\vec{r}_q$, $q = 1, ..., N$ in the image plane. The goal is to find a smooth motion field $\vec{v}$ defined at all positions $\vec{r}$ in the image domain, estimated from the local motion measurements. The motion field $\vec{v}$ can be thought of as an interpolation of the data which obeys a slowness and smoothness prior and which agrees approximately with the local motion measurements. Recall that the visual system can only observe the local motion in the directions $\vec{n}_q = \frac{\vec{u}_q}{|\vec{u}_q|}$ (sometimes called component motion) because of the aperture problem. Hence approximate agreement with local measurements reduces to the constraints:

$$\vec{v}(\vec{r}_q) \cdot \vec{n}_q - \vec{u}_q \cdot \vec{n}_q \approx 0.$$

As illustrated in figure 3, we consider three motion prior terms which quantify the preference for slowness, first-order smoothness and second-order smoothness respectively. Let $\Omega$ denote the image domain – i.e. the set of points $\vec{r} = (r_1, r_2) \in \Omega$. We define the prior to be a Gibbs distribution with energy function of form:

$$E(\vec{v}) = \int_{\Omega} (\frac{\lambda}{\alpha}|\vec{v}|^{\alpha} + \frac{\mu}{\beta}|\nabla\vec{v}|^{\beta} + \frac{\eta}{\gamma}|\triangle\vec{v}|^{\gamma})d\vec{r},$$

where $\lambda, \mu, \eta, \alpha, \beta, \gamma$ are positive parameters and

$$|\vec{v}| = \sqrt{(v_1)^2 + (v_2)^2}, \quad |\nabla \vec{v}| = \sqrt{\left(\frac{\partial v_1}{\partial r_1}\right)^2 + \left(\frac{\partial v_1}{\partial r_2}\right)^2 + \left(\frac{\partial v_2}{\partial r_1}\right)^2 + \left(\frac{\partial v_2}{\partial r_2}\right)^2},$$

$$|\triangle \vec{v}| = \sqrt{\left(\frac{\partial^2 v_1}{\partial r_1^2}\right)^2 + \left(\frac{\partial^2 v_1}{\partial r_2^2}\right)^2 + \left(\frac{\partial^2 v_2}{\partial r_1^2}\right)^2 + \left(\frac{\partial^2 v_2}{\partial r_2^2}\right)^2}.$$

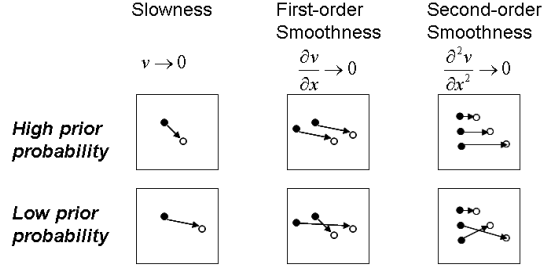

Figure 3: An illustration of three prior terms: (i) slowness, (ii) first-order smoothness, and (iii) second-order smoothness

The (negative log) likelihood function for grating stimuli imposes the measurement constraints and is of form:

$$E(\vec{u}|\vec{v}) = \sum_{q=1}^{N} |\vec{v}(\vec{r}_q) \cdot \vec{n}_q - \vec{u}_q \cdot \vec{n}_q|^p = \sum_{q=1}^{N} |\vec{v}(\vec{r}_q) \cdot \vec{n}_q - |\vec{u}_q||^p.$$

The combined energy function to be minimized is:

$$\inf_{\vec{v}} \left\{ F(\vec{v}) = \frac{c}{p} E(\vec{u}|\vec{v}) + E(\vec{v}) \right\}.$$

This energy is a convex function provided the exponents satisfy $\alpha, \beta, \gamma, p \geq 1$. Therefore the energy minimum can be found by imposing the first order optimality conditions, $\frac{\partial F(\vec{v})}{\partial \vec{v}} = 0$ (the Euler-Lagrange equations). Below we computer these Euler-Lagrange partial differential equations in $\vec{v} = (v_1, v_2)$. We fix the likelihood term by setting $p = 2$ (the exponent of the likelihood term). If $\alpha, \beta, \gamma \neq 2$, the Euler-Lagrange equations are non-linear partial differential equations (PDEs) and explicit solutions cannot be found (if $\alpha, \beta, \gamma = 2$ the Euler-Lagrange equations will be linear and so can be solved by Fourier transforms or Green's functions, as previously done in [6]). To solve these non-linear PDEs we discretize them by finite differences and use iterative gradient descent (i.e. we apply the dynamics $\frac{\partial \vec{v}(\vec{r},t)}{\partial t} = -\frac{\partial F(\vec{v}(\vec{r},t))}{\partial \vec{v}(\vec{r},t)}$ until we reach a fixed state). More precisely, we initialize $\vec{v}(\vec{r}, 0)$ at random, and solve the update equation for $t > 0$:

$$\frac{\partial v_k}{\partial t}(\vec{r}, t) = -\lambda |\vec{v}|^{\alpha-2} v_k + \mu \mathrm{div}\left(|\nabla \vec{v}|^{\beta-2} \nabla v_k\right) - \eta \triangle\left(|\triangle \vec{v}|^{\gamma-2} \triangle v_k\right)$$
$$- c\left(\vec{v}(\vec{r}_q) \cdot \vec{n}_q - \vec{u}_q \cdot \vec{n}_q\right)^{p-1} n_{k_q} \delta_{\vec{r}, \vec{r}_q},$$

where $k = 1, 2$, $\delta_{\vec{r}, \vec{r}_q} = 1$ if $\vec{r} = \vec{r}_q$ and $\delta_{\vec{r}, \vec{r}_q} = 0$ if $\vec{r} \neq \vec{r}_q$. Since the powers $\alpha - 2$, $\beta - 2$, $\gamma - 2$ become negative when the positive exponents $\alpha, \beta, ...$ take value 1, we include a small $\epsilon = 10^{-6}$ inside the square roots to avoid division by zero (when calculating terms like $|.|$). The algorithm stops when the difference between two consecutive energy estimates is close to zero (i.e. the stopping criterion is based on thresholding the energy change).

Our implementation discretized the Euler-Lagrange equations, as specified below. Let $\vec{B}^{(l)} = |\nabla \vec{v}^{(l)}|^{\beta-2}$, $\vec{C}^{(l)} = |\triangle \vec{v}^{(l)}|^{\gamma-2}$, $\vec{A}^{(l)} = |\vec{v}^{(l)}|^{\alpha-2}$, where $l$ denotes time discretization with $\triangle t$ the time-step, and $(i, j)$ denotes space discretization with $h = \triangle r_1 = \triangle r_2$ being the space-step. Then the above PDE's can be discretized as

$$\frac{v_{k_{i,j}}^{(l+1)} - v_{k_{i,j}}^{(l)}}{\triangle t} = Fid_{k_{i,j}}^{(l)} - \lambda \vec{A}_{i,j} v_{k_{i,j}}^{(l+1)}$$

$$
\begin{aligned}
+\quad & \frac{\mu}{h^2}[(-\vec{B}_{i,j-1}^{(l)} - \vec{B}_{i-1,j}^{(l)} - 2\vec{B}_{i,j}^{(l)})v_{k_{i,j}}^{(l+1)} \\
+\quad & \vec{B}_{i,j}^{(l)}v_{k_{i+1,j}}^{(l)} + \vec{B}_{i-1,j}^{(l)}v_{k_{i-1,j}}^{(l)} + \vec{B}_{i,j}^{(l)}v_{k_{i,j+1}}^{(l)} + \vec{B}_{i,j-1}^{(l)}v_{k_{i,j-1}}^{(l)}] \\
-\quad & \frac{\eta}{h^4}\{(\vec{C}_{i+1,j}^{(l)} + \vec{C}_{i-1,j}^{(l)} + 16\vec{C}_{i,j}^{(l)} + \vec{C}_{i,j+1}^{(l)} + \vec{C}_{i,j-1}^{(l)})v_{k_{i,j}}^{(l+1)} \\
-\quad & 4[(\vec{C}_{i+1,j}^{(l)} + \vec{C}_{i,j}^{(l)})v_{k_{i+1,j}}^{(l)} + (\vec{C}_{i-1,j}^{(l)} + \vec{C}_{i,j}^{(l)})v_{k_{i-1,j}}^{(l)} \\
+\quad & (\vec{C}_{i,j+1}^{(l)} + \vec{C}_{i,j}^{(l)})v_{k_{i,j+1}}^{(l)} + (\vec{C}_{i,j-1}^{(l)} + \vec{C}_{i,j}^{(l)})v_{k_{i,j-1}}^{(l)}] \\
+\quad & (\vec{C}_{i+1,j}^{(l)} + \vec{C}_{i,j+1}^{(l)})v_{k_{i+1,j+1}}^{(l)} + (\vec{C}_{i+1,j}^{(l)} + \vec{C}_{i,j-1}^{(l)})v_{k_{i+1,j-1}}^{(l)} \\
+\quad & (\vec{C}_{i-1,j}^{(l)} + \vec{C}_{i,j+1}^{(l)})v_{k_{i-1,j+1}}^{(l)} + (\vec{C}_{i-1,j}^{(l)} + \vec{C}_{i,j-1}^{(l)})v_{k_{i-1,j-1}}^{(l)} \\
+\quad & \vec{C}_{i+1,j}^{(l)}v_{k_{i+2,j}}^{(l)} + \vec{C}_{i-1,j}^{(l)}v_{k_{i-2,j}}^{(l)} + \vec{C}_{i,j+1}^{(l)}v_{k_{i,j+2}}^{(l)} + \vec{C}_{i,j-1}^{(l)}v_{k_{i,j-2}}^{(l)}\}
\end{aligned}
$$

where $Fid_{k_{i,j}} = \begin{cases} -c\left(\vec{v}(\vec{r}_q)\cdot\vec{n}_q - \vec{u}_q\cdot\vec{n}_q\right)^{p-1} n_{kq} & \text{if } \vec{r}_q = (i,j) \\ 0 & \text{otherwise} \end{cases}$ . Letting

$$
\vec{E1}_{i,j} = \vec{B}_{i,j-1} + \vec{B}_{i-1,j} + 2\vec{B}_{i,j}, \ \vec{E2}_{i,j} = \vec{C}_{i+1,j} + \vec{C}_{i-1,j} + 16\vec{C}_{i,j} + \vec{C}_{i,j+1} + \vec{C}_{i,j-1},
$$
$$
\vec{E3}_{i,j} = \vec{C}_{i+1,j} + \vec{C}_{i,j}, \ \vec{E4}_{i,j} = \vec{C}_{i-1,j} + \vec{C}_{i,j}, \ \vec{E5}_{i,j} = \vec{C}_{i,j+1} + \vec{C}_{i,j}, \ \vec{E6}_{i,j} = \vec{C}_{i,j-1} + \vec{C}_{i,j},
$$
$$
\vec{E7}_{i,j} = \vec{C}_{i+1,j} + \vec{C}_{i,j+1}, \ \vec{E8}_{i,j} = \vec{C}_{i+1,j} + \vec{C}_{i,j-1}, \ \vec{E9}_{i,j} = \vec{C}_{i-1,j} + \vec{C}_{i,j+1},
$$
$$
\vec{E10}_{i,j} = \vec{C}_{i-1,j} + \vec{C}_{i,j-1}, \ \vec{E11} = 1/(1 + \triangle t(\lambda\vec{A} + \frac{\mu}{h^2}\vec{E1} + \frac{\eta}{h^4}\vec{E2})),
$$

we can solve for $v^{(l+1)}$ and we obtain

$$
\begin{aligned}
v_{k_{i,j}}^{(l+1)} \quad = \quad & \vec{E11}_{i,j}^{(l)}\Big(v_{k_{i,j}}^{(l)} + \triangle t\{Fid_{k_{i,j}}^{(l)} + \frac{\mu}{h^2}(\vec{B}_{i,j}^{(l)}v_{k_{i+1,j}}^{(l)} + \vec{B}_{i-1,j}^{(l)}v_{k_{i-1,j}}^{(l)} + \vec{B}_{i,j}^{(l)}v_{k_{i,j+1}}^{(l)} + \vec{B}_{i,j-1}^{(l)}v_{k_{i,j-1}}^{(l)}) \\
-\quad & \frac{\eta}{h^4}[-4(\vec{E3}_{i,j}^{(l)}v_{k_{i+1,j}}^{(l)} + \vec{E4}_{i,j}^{(l)}v_{k_{i-1,j}}^{(l)} + \vec{E5}_{i,j}^{(l)}v_{k_{i,j+1}}^{(l)} + \vec{E6}_{i,j}^{(l)}v_{k_{i,j-1}}^{(l)}) \\
+\quad & \vec{E7}_{i,j}^{(l)}v_{k_{i+1,j+1}}^{(l)} + \vec{E8}_{i,j}^{(l)}v_{k_{i+1,j-1}}^{(l)} + \vec{E9}_{i,j}^{(l)}v_{k_{i-1,j+1}}^{(l)} + \vec{E10}_{i,j}^{(l)}v_{k_{i-1,j-1}}^{(l)} \\
+\quad & \vec{C}_{i+1,j}^{(l)}v_{k_{i+2,j}}^{(l)} + \vec{C}_{i-1,j}^{(l)}v_{k_{i-2,j}}^{(l)} + \vec{C}_{i,j+1}^{(l)}v_{k_{i,j+2}}^{(l)} + \vec{C}_{i,j-1}^{(l)}v_{k_{i,j-2}}^{(l)}]\}\Big).
\end{aligned}
$$

## 4   Experiments

We compared two possible functional forms for the motion prior: (1) the Laplace distribution with L1-norm regularization, with $\alpha = \beta = \gamma = 1$, (2) the Gaussian distribution with L2-norm regularization, with $\alpha = \beta = \gamma = 2$. Since the main goal of this work is to discover motion priors, we employed the same likelihood term with $p = 2$ for both models. We used the performance of human subjects in the first experimental session to estimate the weights of the three prior terms, $\lambda, \mu, \eta$, for each functional form. We then validated the predictions of the model by comparing them with human performance in a second experimental session which uses different stimulus parameters.

### 4.1   Stimulus

We used a multiple-aperture stimulus [13] which consists of 12 by 12 drifting sine-wave gratings within a square window subtending 8°. Each element (0.5°) was composed of an oriented sinusoidal grating of 5.6 cycles/deg spatial frequency, which was within a stationary Gaussian window. The contrast of the elements was 0.2. The motion stimulus included 20 time frames which were presented within 267 ms. The global motion stimulus was generated as follows. First, the orientation of each local grating element was randomly determined. Second, a global motion (also called 2D motion, with the speed of 1 deg/sec) direction was chosen. Third, a certain proportion of elements (signal elements) were assigned with the predetermined 2D motion , while each of the remaining elements (noise elements) was assigned a random 2D motion. Finally, with its orientation and 2D motion velocity, the drifting speed for each element was computed so that the local (or component) drifting velocity was consistent with the assigned 2D motion velocity. As shown in figure 4 the global motion strength was controlled by varying the proportion of signal elements in the stimulus (i.e., the

coherence ratio). Stimuli with high ratio exhibited more coherent motion, and stimuli with low ratio exhibited more random motion.

In all the experiments reported in this paper, each participant completed two experiment sessions with different stimulus parameters. The goal of session 1 was parameter estimation: to estimate the weights of the three prior terms – slowness, first-order smoothness and second-order smoothness, – for each model. Session 2 was for model validation: using the weights estimated from session 1 to predict subject performance for different experimental conditions.

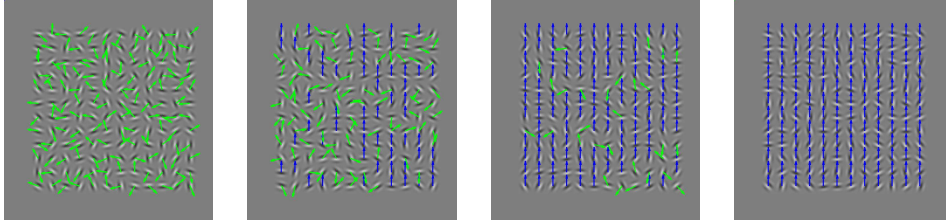

Figure 4: Stimulus illustration. Multiple-aperture stimuli with coherence ratio of 0, 0.4, 0.8 and 1 from left to right. the blue and green arrows indicate the 2D motion directions assigned for signal and noise elements, respectively.

## 4.2   Experiment 1

### 4.2.1   Procedure

There were two separate sessions in Experiment 1. On each trial of the first session, observers were presented with two motion patterns, one after another. The first one was the reference motion pattern, which always moved upward (0 degree), and the second one was the test motion pattern, whose global motion direction was either tilted towards the left or the right relative to the reference pattern. Both patterns lasted for 267 ms with 500 ms inter-stimulus interval. The observer's task was to determine whether the global motion direction of the test pattern was more towards the left or right relative to the reference pattern. In order to make sure observers understood the task and were able to perceive the global motion, before the beginning of the first session, observers passed a test session in which they achieved 90% accuracy in 40 consecutive trials with 80% coherence and 20 (or 45) degrees of angular difference. To allow observers to familiarize themselves with the task, before each experimental session observers went through a practice session with 10 blocks of 25 trials.

The first session consisted of 20 blocks of 50 trials. the coherence ratio was constant within each block. The observer's discrimination performance was measured for ten coherence ratios (0, 0.1, 0.2, .., 0.9) in the first session. The angular difference between the reference and test motion was fixed for each observer in the entire session (2 degrees for observers AL, MW and AE; 45 degrees for OQ and CC). The second session was identical to the first one, except that the coherence ratio was fixed at 0.7, and the angular difference between the global motion directions of the reference and the test patterns was varied across blocks (ten angular differences: 1, 5, 10, .., 45 degrees).

### 4.2.2   Results

We implemented motion models with the Laplace prior distribution (termed "L1 model") and the Gaussian prior (termed "L2 model"). As the first step, exhaustive search was conducted to find a set of weights for the prior terms that provided the best fit to the human psychometric performance in experimental session 1. Table 1 reports the estimated parameters for each individual subject using the L1 and L2 models. There was clear individual difference for the estimated weight values. However, across all five subjects, large weight values were found for the second-order smoothness terms, indicating the contribution from higher-order smoothness preference is important in perceiving global motion from multiple-aperture stimulus.

Figure 5 shows the results from each individual participant and best-fitting model performance. The results clearly show the L1 model provided the better fit to human data when compared to the L2 model. In general,humans appear to be sensitive to the inclusion of noise elements, and perform

Table 1: Estimated weights $\lambda, \mu, \eta$ of slowness, first-order smoothness and second-order smoothness prior terms, for L1 and L2-norm model

| Subjects | L1 $\lambda$ | L1 $\mu$ | L1 $\eta$ | L2 $\lambda$ | L2 $\mu$ | L2 $\eta$ |
|---|---|---|---|---|---|---|
| AE | 0.001 | 1 | 15000 | 0.01 | 100 | 16000 |
| AL | 0.01 | 100 | 16000 | 0.01 | 1 | 16000 |
| CC | 0.001 | 0.1 | 16000 | 0.001 | 0.1 | 16000 |
| MW | 0.001 | 10 | 17000 | 0.01 | 1 | 20000 |
| OQ | 0.01 | 100 | 18000 | 0.01 | 100 | 18000 |

worse than the L2 model, which tends to strongly encourage smoothness over the entire display window.

In experimental session 2, the two models predicted performance as a function of angular difference between the reference motion and the test motion. As shown in figure 7, the L1 model yielded less error in fitting human performance than did the L2 model. This result illustrates the power of the L1 model in predicting human performance in motion tasks different from the tasks used for estimating model parameters.

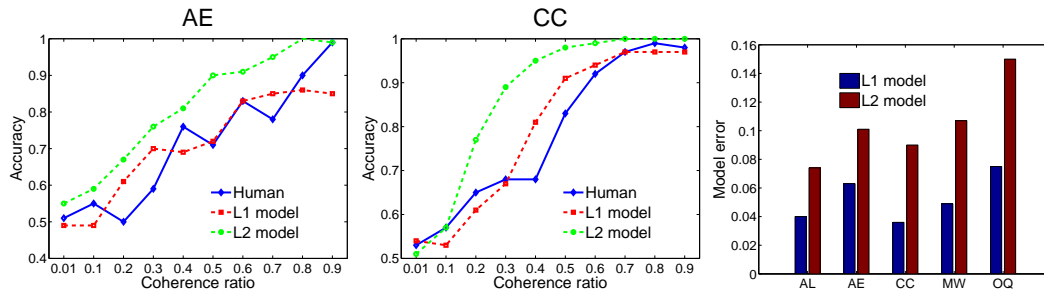

Figure 5: Comparison between human performance and model predictions in session 1. Left two plots, accuracy as a function of *coherence ratio* for two representative subjects. Blue solid lines indicate human performance. Red and green dashed lines indicate L1 and L2 model predictions with the best fitted parameters. Right plot, model error for all five subjects. The model error was computed as the mean absolute difference between human performance and model predictions. L1 model consistently fits human performance better than L2 model for all subjects

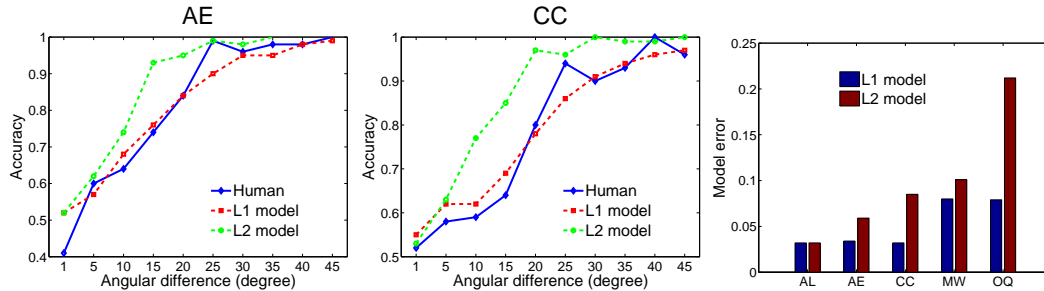

Figure 6: Comparison between human performance and model predictions in session 1. Left two plots, accuracy as a function of *angular difference between the reference and the test motion* for two representative subjects. Blue solid lines indicate human performance. Red and Green dashed lines indicate L1 and L2 model predictions. Right plot, model error for all five subjects. Less errors from L1 model indicate that L1 model consistently fits human performance better than L2 model for all subjects

## 4.3 Experiment 2

The results of Experiment 1 clearly support the conclusion that the motion model with Laplace prior (L1-norm regularization) fits human performance better than does the model with Gaussian prior

(L2 model). In Experiment 2, we compared human motion judgment with predictions of the L1 model on each trial, rather than using the average performance as in Experiment 1. Such a detailed comparison can provide quantitative measures of how well the L1 model is able to predict human motion judgment for specific stimuli.

In Experiment 2, the first session was identical to that in Experiment 1, in which angular difference in the two global motion directions were fixed (45 degrees for all observers) while the coherence ratio was varied. In the second session, observers were presented with one motion stimulus on each trial. The global motion direction of the pattern was randomly selected from 24 possible directions (with a 15-degree difference between two adjacent directions). Observers reported their perceived global motion directions by rotating a line after the motion stimulus disappeared from the screen. The experiment included 12 blocks (each with 48 trials) and six coherence ratios (0, 0.1, 0.3, .., 0.9). A two-pass design was used to let each observer run the identical session twice in order to measure the reliability of the observer's judgments.

We used human performance in session 1 to estimate model parameters: weights $\lambda, \mu, \eta$ for slowness, first-order smoothness and second-order smoothness prior terms for each individual participant. Since identical stimuli were used in the two runs of session 2, we can quantify the reliability of the observer's judgment by computing the response correlation across trials in these two runs. As shown in the left plot of figure 7, human observers' responses were significantly correlated in the two runs, even in the condition of **random motion** (coherence ratio is close to 0). The correlated responses in these subthreshold conditions suggest that human observers are able to provide consistent interpretation of motion flow, even when the motion is random. The right plot of figure 7 shows the trial-by-trial correlation between human motion judgments with model-predicted global motion direction. The model-human correlations were comparable to human self-correlations. Even in the random motion condition (where the coherence ratio is 0), the correlation between the model and human judgments is greater than 0.5, indicating the predictive power of the model. We also noticed that the correlation between human and L2 model was around 8 percent worse than the human self-correlation and the correlation between the L1 model and humans. This finding further demonstrated that the L1 model provided a better fit to human data than did the L2 model.

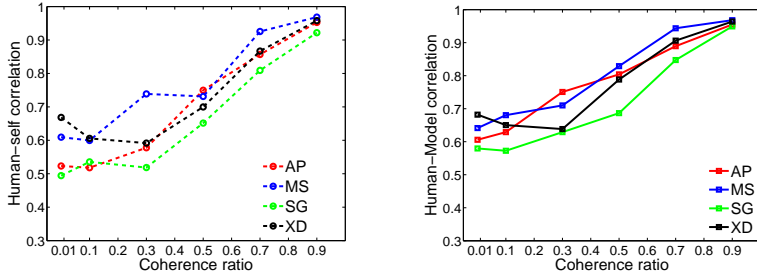

Figure 7: Comparison between human performance and model predictions using trial-by-trial correlation. Left plot, human self correlation between two runs of identical experimental sessions. Right plot, correlation between human motion judgement and model predicted global motion direction. The significant correlation between human and the model indicates the L1 model is able to predict human motion judgment for specific stimuli, even in the random display, i.e., coherence ratio close to 0.

## 5  Conclusions

We found that a motion prior in the form of the Laplace distribution with L1-norm regularization provided significantly better agreement with human performance than did Gaussian priors with L2-norm. We also showed that humans weighted second-order motion smoothness much higher than first-order smoothness and slowness. Furthermore, model predictions using this Laplace prior were consistent with human perception of coherent motion, even for random displays. Overall our results suggest that human motion perception for these types of stimuli can be well modeled using Laplace priors.

**Acknowledgments**

This research was supported by NSF grants IIS-613563 to AY and BCS-0843880 to HL.

# References

[1] R. Sekuler, S.N.J. Watamaniuk and R. Blake. Perception of Visual Motion. In Steven's Handbook of Experimental Psychology. Third edition. H. Pashler, series editor. S. Yantis, volume editor. J. Wiley Publishers. New York. 2002.

[2] L. Welch. The perception of moving plaids revewals two processing stages. *Nature*,337,734-736. 1989.

[3] P. Schrater, D. Knill and E. Simoncelli. Mechanisms of visual motion detection. *Nature Neuroscience*, 3, 64-68. 2000.

[4] J. A. Movhson and W. T. Newsome. Visual response properties of striate cortical neurons projecting to area MT in macaque monkeys. *Visual Neuroscience*, 16, 7733-7741. 1996.

[5] N. C. Rust, V. Mante, E. P. Simoncelli and J. A. Movshon. How MT cells analyze the motion of visual patterns. *Nature Neuroscience*, 9(11), 1421-1431. 2006.

[6] A.L. Yuille and N.M. Grzywacz. A computational theory for the perception of coherent visual motion. *Nature*, 333,71-74. 1988.

[7] A.L. Yuille and N.M. Grzywacz. A Mathematical Analysis of the Motion Coherence Theory. *International Journal of Computer Vision*. 3. pp 155-175. 1989.

[8] Y. Weiss, E.P. Simoncelli, and E.H. Adelson. Motion illusions as optimal percepts. *Nature Neuroscience*, 5, 598-604. 2002.

[9] H. Lu and A.L. Yuille. Ideal Observers for Detecting Motion: Correspondence Noise. *Advances in Neural Information Processing Systems 7*, pp. 827-834. 2005.

[10] A.A. Stocker and E.P. Simoncelli. Noise characteristics and prior expectations in human visual speed perception. *Nature Neuroscience*, 9(4), pp. 578-585, 2006.

[11] S. Roth and M. J. Black. On the spatial statistics of optical flow. *International Journal of Computer Vision*, 74(1), pp. 33-50, 2007.

[12] C. Liu, W. T. Freeman, E. H. Adelson and Y. Weiss. *IEEE Conference on Computer Vision and Pattern Recognition*, 2008.

[13] Amano, K., Edwards, M., Badcock, D. R. and Nishida, S. Adaptive pooling of visual motion signals by the human visual system revealed with a novel multi-element stimulus. *Journal of Vision*, 9(3), 4, 1-25, 2009.

